# Visualizing Group Structure*

**Marcus Held, Jan Puzicha, and Joachim M. Buhmann**
Institut für Informatik III,
Römerstraße 164, D-53117 Bonn, Germany
email: {held,jan,jb}.cs.uni-bonn.de,
WWW: `http://www-dbv.cs.uni-bonn.de`

## Abstract

Cluster analysis is a fundamental principle in exploratory data analysis, providing the user with a description of the group structure of given data. A key problem in this context is the interpretation and visualization of clustering solutions in high–dimensional or abstract data spaces. In particular, probabilistic descriptions of the group structure, essential to capture inter–cluster relationships, are hardly assessable by simple inspection of the probabilistic assignment variables. We present a novel approach to the visualization of group structure. It is based on a statistical model of the object assignments which have been observed or estimated by a probabilistic clustering procedure. The objects or data points are embedded in a low dimensional Euclidean space by approximating the observed data statistics with a Gaussian mixture model. The algorithm provides a new approach to the visualization of the inherent structure for a broad variety of data types, e.g. histogram data, proximity data and co–occurrence data. To demonstrate the power of the approach, histograms of textured images are visualized as an example of a large–scale data mining application.

## 1  Introduction

Clustering and visualization are key issues in exploratory data analysis and are fundamental principles of many unsupervised learning schemes. For a given data set, the aim of any clustering approach is to extract a description of the inherent group structure. The object space is partitioned into groups where each partition

is as homogeneous as possible and two partitions are maximally heterogeneous. For several reasons it is useful to deal with probabilistic partitioning approaches:

1. The data generation process itself might be stochastic, resulting in overlapping partitions. Thus, a probabilistic group description is adequate and provides additional information about the inter–cluster relations.
2. The number of clusters might be chosen too large. Forcing the algorithm to a hard clustering solution creates artificial structure not supported by the data. On the other hand, superfluous clusters can be identified by a probabilistic group description .
3. There exists theoretical and empirical evidence that probabilistic assignments avoid over-fitting phenomena [7].

Several well-known clustering schemes result in fuzzy cluster assignments: For the most common type of vector–valued data, heuristic fuzzy clustering methods were suggested [4, 5]. In a more principled way, *deterministic annealing* algorithms provide fuzzy clustering solutions for a given cost function with a rigorous statistical foundation and have been developed for vectorial [9], proximity [6] and histogram data [8]. In mixture model approaches the assignments of objects to groups are interpreted as missing data. Its conditional expectations given the data and the estimated cluster parameters are computed during the E–step in the corresponding EM–algorithm and can be understood as assignment probabilities.

The aim of this contribution is to develop a generic framework to visualize such probabilities as distances in a low dimensional Euclidean space. Especially in high dimensional or abstract object spaces, the interpretation of fuzzy group structure is rather difficult, as humans do not perform very well in interpreting probabilities. It is, therefore, a key issue to make an interpretation of the cluster structure more feasible. In contrast to multidimensional scaling (MDS), where objects are embedded in low dimensional Euclidean spaces by preserving the original inter object distances [3], our approach yields a mixture model in low dimensions, where the probabilities for assigning objects to clusters are maximally preserved. The proposed approach is similar in spirit to data visualization methods like projection pursuit clustering, GTM [1], simultaneous clustering and embedding [6], and hierarchical latent variable models [2]. It also aims on visualizing high dimensional data. But while the other methods try to model the data itself by a low dimensional generator model, we seek to model the inferred probabilistic grouping structure. As a consequence, the framework is generic in the sense that it is applicable to any probabilistic or fuzzy group description.

The key idea is to interpret a given probabilistic group description as an observation of an underlying random process. We estimate a low–dimensional statistical model by maximum likelihood inference which provides the visualization. To our knowledge the proposed algorithm provides the first solution to the visualization of distributional data, where the observations of an object consists of a histogram of measured features. Such data is common in data mining applications like image retrieval where image similarity is often based on histograms of color or texture features. Moreover, our method is applicable to proximity and co–occurrence data.

## 2   Visualizing Probabilistic Group Structure

Let a set of $N$ (abstract) objects $\mathcal{O} = \{o_1, \ldots, o_N\}$ be given which have been partitioned into $K$ groups or clusters. Let the fuzzy assignment of object $o_i$ to cluster $C_\nu$ be given by $q_{i\nu} \in [0, 1]$, where we assume $\sum_{\nu=1}^{K} q_{i\nu} = 1$ to enable a probabilistic interpretation. We assume that there exists an underlying "true" assignment of

objects to clusters which we encode by Boolean variables $M_{i\nu}$ denoting whether object $o_i$ belongs to (has been generated by) cluster $C_\nu$. We thus interpret $q_{i\nu}$ as an *empirical estimate* of the probability $\mathbf{P}(M_{i\nu} = 1)$. For notational simplicity, we summarize the assignment variables in matrices $\mathbf{Q} = (q_{i\nu})$ and $\mathbf{M} = (M_{i\nu})$.

The key idea for visualizing group structure is to exploit a low–dimensional statistical model which "explains" the observed $q_{i\nu}$. The parameters are estimated by maximum likelihood inference and provide a natural data visualization. Gaussian mixture models in low dimensions (typically $d = 2$ or $d = 3$) are often appropriate but the scheme could be easily extended to other classes, e.g. hierarchical models. To define the Gaussian mixture model, we first introduce a set of prototypes $\mathcal{Y} = \{\mathbf{y}_1, \dots, \mathbf{y}_K\} \subset \mathbb{R}^d$ representing the $K$ clusters, and a set vector–valued object parameters $\mathcal{X} = \{\mathbf{x}_1, \dots, \mathbf{x}_N\} \subset \mathbb{R}^d$. To model the assignment probabilities, the prototypes $\mathcal{Y}$ and the data points $\mathcal{X}$ are chosen such that the resulting assignment probabilities are maximally similar to the given frequencies $\mathbf{Q}$. For the Gaussian mixture model we have

$$\mathbf{P}\left(\mathbf{M}|\mathcal{X}, \mathcal{Y}\right) = \prod_{i=1}^{N} \prod_{\nu=1}^{K} (m_{i\nu})^{M_{i\nu}}, \text{ with } m_{i\nu} = \frac{\exp\left(-\beta\|\mathbf{x}_i - \mathbf{y}_\nu\|^2\right)}{\sum_{\mu=1}^{K} \exp\left(-\beta\|\mathbf{x}_i - \mathbf{y}_\mu\|^2\right)} \ . \tag{1}$$

Note that the probability distribution is invariant under translation and rotation of the complete parameter sets $\mathcal{X}, \mathcal{Y}$. In addition, the scale parameter $\beta$ could be dropped since a change of $\beta$ only results in a rescaling of the prototypes $\mathcal{Y}$ and the data points $\mathcal{X}$. For the observation $\mathbf{Q}$ the log–likelihood is given by[1]

$$\mathcal{L}_{\mathbf{Q}}\left(\mathcal{X}, \mathcal{Y}\right) = \sum_{i=1}^{N} \sum_{\nu=1}^{K} q_{i\nu} \log m_{i\nu} \ . \tag{2}$$

It is worth to note that when the $q_{i\nu} = \langle M_{i\nu} \rangle_{\mathbf{P}^{\text{true}}}$ are estimates obtained by a factorial distribution, i.e. $\mathbf{P}^{\text{true}}(\mathbf{M}) = \prod_i \sum_\nu M_{i\nu} q_{i\nu}$, then maximizing (2) is identical to minimizing the Kullback–Leibler (KL–)divergence $D_{\text{KL}}(\mathbf{P}^{\text{true}}\|\mathbf{P}) = \sum_{\mathbf{M}} \mathbf{P}^{\text{true}} \log(\mathbf{P}^{\text{true}}/\mathbf{P})$. In that case the similarity to the recent approach of Hofmann et al. [6] proposed as the minimization of $D_{\text{KL}}(\mathbf{P}\|\mathbf{P}^{\text{true}})$ becomes apparent. Compared to [6] the role of $\mathbf{P}$ and $\mathbf{P}^{\text{true}}$ is interchanged. From an information–theoretic viewpoint $D_{\text{KL}}(\mathbf{P}^{\text{true}}\|\mathbf{P})$ is a better choice as it quantifies the coding inefficiency of assuming the distribution $\mathbf{P}$ when the true distribution is $\mathbf{P}^{\text{true}}$. Note that the choice of the KL–divergence as a distortion measure for distributions follows intrinsically from the likelihood principle. Maximum likelihood estimates are derived by differentiation:

$$\frac{\partial \mathcal{L}_{\mathbf{Q}}}{\partial \mathbf{x}_i} = \sum_{\nu=1}^{K} \frac{q_{i\nu}}{m_{i\nu}} \frac{\partial m_{i\nu}}{\partial \mathbf{x}_i} = -2\beta \sum_{\nu=1}^{K} q_{i\nu} \left(\sum_{\mu=1}^{K} m_{i\mu} \mathbf{y}_\mu - \mathbf{y}_\nu\right) \ , \tag{3}$$

$$\frac{\partial \mathcal{L}_{\mathbf{Q}}}{\partial \mathbf{y}_\alpha} = \sum_{i=1}^{N} \sum_{\nu=1}^{K} \frac{q_{i\nu}}{m_{i\nu}} \frac{\partial m_{i\nu}}{\partial \mathbf{y}_\alpha} = -2\beta \sum_{i=1}^{N} \sum_{\nu=1}^{K} q_{i\nu} (m_{i\alpha} - \delta_{\alpha\nu}) (\mathbf{x}_i - \mathbf{y}_\alpha)$$

$$= -2\beta \sum_{i=1}^{N} (m_{i\alpha} - q_{i\alpha}) (\mathbf{x}_i - \mathbf{y}_\alpha) \ . \tag{4}$$

The gradients can be used for any gradient descent scheme. In the experiments, we used (3)–(4) in conjunction with a simple gradient descent technique, which has

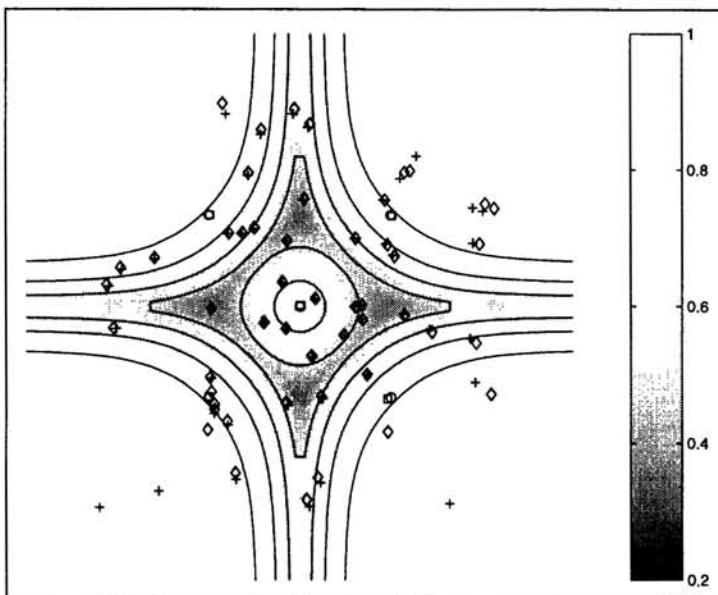

Figure 1: Visualization of two–dimensional artificial data. Original data generated by the mixture model with $\beta = 1.0$ and 5 prototypes. Crosses denote the data points $x_i$, circles the prototypes $y_\alpha$. The embedding prototypes are plotted as squares, while the embedding data points are diamonds. The contours are given by $f(x) = \max_\alpha \left( \exp\left(-\beta||\mathbf{x} - \mathbf{y}_\alpha||^2\right)/\sum_{\mu=1}^{K} \exp\left(-\beta||\mathbf{x} - \mathbf{y}_\mu||^2\right) \right)$. For visualization purposes the embedding is translated and rotated in the correct position.

been observed to be efficient and reliable up to a few hundred objects. From (4) an explicit formula for the prototypes may be recovered

$$\mathbf{y}_\alpha = \sum_{i=1}^{N} (m_{i\alpha} - q_{i\alpha})\, \mathbf{x}_i \bigg/ \sum_{i=1}^{N} (m_{i\alpha} - q_{i\alpha}) \tag{5}$$

which can be interpreted as an alternative centroid rule. The position of the prototypes is dominated by objects with a large deviation between modeled and measured assignment probabilities. Note that (5) should not be used as an iterative equation as the corresponding fixed point is not contractive.

## 3 Results

As a first experiment we discuss the approximately recoverable case, where we sample from (1) to generate artificial two–dimensional data and infer the positions of the sample points and of the prototypes by the visualizing group structure approach (see Fig. 1). Due to iso–contour lines in the generator density and in the visualization density not all data positions are recovered exactly. We like to emphasize that the complete information available on the grouping structure of the data is preserved, since the mean KL–divergence is quite small ($\approx 2.10 \cdot 10^{-5}$). It is worth mentioning that the rank–order of the assignments of objects $i$ to clusters $\alpha$ is completely preserved.

For many image retrieval systems image similarity has been defined as similarity of occurring feature coefficients, e.g. colors or texture features. In [7], a novel statistical mixture model for distributional data, the probabilistic histogram clustering (ACM), has been proposed which we applied to extract the group structure inherent in image databases based on histograms of textured image features. The ACM explains the observed data by the generative model:

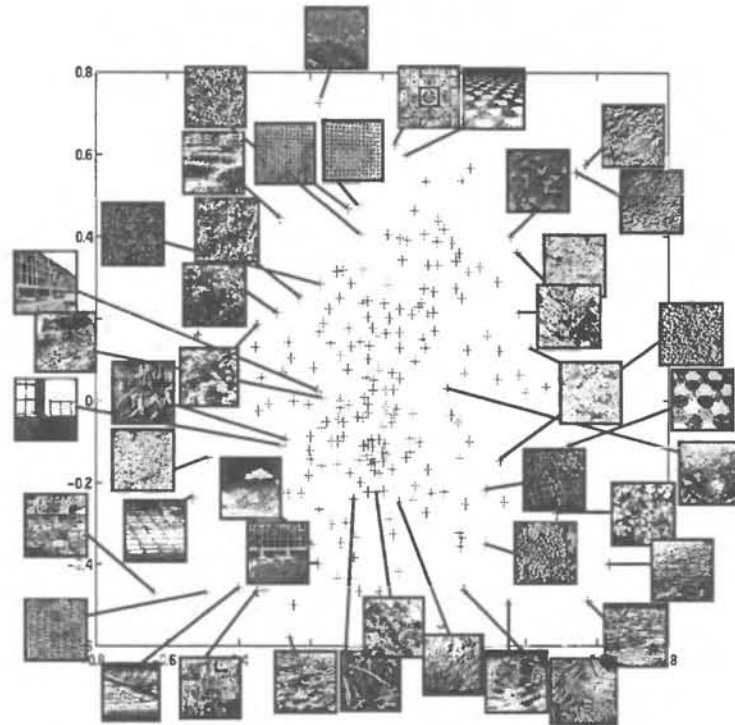

Figure 2: Embedding of the VisTex database with MDS.

1. select an object $o_i \in \mathcal{O}$ with probability $p_i$,
2. choose a cluster $\mathcal{C}_\alpha$ according to the cluster membership $M_{i\alpha}$ of $o_i$,
3. sample a feature $v_j \in \mathcal{V}$ from the cluster conditional distribution $q_{j|\alpha}$.

This generative model is formalized by

$$P\left(o_i, v_j | \mathbf{M}, p, q\right) = p_i \sum_{\alpha=1}^{K} M_{i\alpha} q_{j|\alpha} \ . \tag{6}$$

The parameters are estimated by maximum likelihood inference. The assignments $M_{i\alpha}$ are treated as unobserved data in an (annealed) EM procedure, which provides a probabilistic group description. For the details we refer to [7].

In the experiments, texture features are extracted by a bank of 12 Gabor filters with 3 scales and 4 orientations. Different Gabor channels are assumed to be independently distributed, which results in a concatenated histogram of the empirically measured channel distributions. Each channel was discretized into 40 bins resulting in a 480 dimensional histogram representing one image. For the experiments two different databases were used.

In Fig. 3 a probabilistic $K = 10$ cluster solution with 160 images containing different textures taken from the Brodatz album is visualized. The clustering algorithm produces 8 well separated clusters, while the two clusters in the mid region exhibit substantial overlap. A close inspection of these two clusters indicates that the fuzziness of the assignments in this area is plausible as the textures in this area have similar frequency components in common.

The result for a more complex database of 220 textured images taken from the MIT VisTex image database with a large range of uniformly and non-uniformly textured images is depicted in Fig. 4. This plot indicates that the proposed approach provides a structured view on image databases. Especially the upper left cluster yields some

insight in the clustering solution, as this cluster consists of a large range of non-uniformly textured images, enabling the user to decide that a higher number of clusters might yield a better solution. The visualization approach fits naturally in an interactive scenario, where the user can choose interactively data points to focus his examination to certain areas of interest in the clustering solution.

For comparison, we present in Fig. 2 a multidimensional scaling (Sammon's mapping [3]) solution for the VisTex database. A detailed inspection of this plot indicates, that the embedding is locally quiet satisfactory, while no global structure of the database is visible. This is explained by the fact, that Sammon's mapping only tries to preserve the object distances, while our novel approach first extracts group structure in a high dimensional feature space and than embeds this group structure in a low dimensional Euclidean space. While MDS completely neglects the grouping structure we do not care for the exact inter object distances.

## 4   Conclusion

In this contribution, a generic framework for the low-dimensional visualization of probabilistic group structure was presented. The effectiveness of this approach was demonstrated by experiments on artificial data as well as on databases of textured images. While we have focussed on histogram data the generality of the approach makes it feasible to visualize a broad range of different data types, e.g. vectorial, proximity or co-occurrence data. Thus, it is useful in a broad variety of applications, ranging from image or document retrieval tasks, the analysis of marketing data to the inspection of protein data. We believe that this technique provides the user substantial insight in the validity of clustering solutions making the inspection and interpretation of large databases more practicable.

A natural extension of the proposed approach leads to the visualization of hierarchical cluster structures by a hierarchy of visualization plots.

## Footnotes

*This work has been supported by the German Research Foundation (DFG) under grant #BU 914/3–1, by the German Israel Foundation for Science and Research Development (GIF) under grant #1-0403-001.06/95 and by the Federal Ministry for Education, Science and Technology (BMBF #01 M 3021 A/4).

[1] Here, it is implicitly assumed that all $q_{i\nu}$ have been estimated based on the same amount of information.

## References

[1] C.M. Bishop, M. Svensén, and C.K.I. Williams. GTM: the generative topographic mapping. *Neural Computation*, 10(1):215–234, 1998.

[2] C.M. Bishop and M. E. Tipping. A hierarchical latent variable model for data visualization. Technical Report NCRG/96/028, Neural Computing Research Group Dept. Of Computer Science & Applied Mathematics, Aston University, 1998.

[3] T.F. Cox and M.A.A. Cox. *Multidimensional Scaling*, volume 59 of *Mongraphs on statistics and applied probability*. Chapman & Hall, London, New York, 1994.

[4] J.C. Dunn. A fuzzy relative of the ISODATA process and its use in detecting well-separated clusters. *Journal of Cybernetics*, 3:32–57, 1975.

[5] I. Gath and A. Geva. Unsupervised optimal fuzzy clustering. *IEEE Transactions on Pattern Analysis and Machine Intelligence*, 11:773–781, 1989.

[6] T. Hofmann and J. M. Buhmann. Pairwise data clustering by deterministic annealing. *PAMI*, 19(1):1–25, 1997.

[7] T. Hofmann, J. Puzicha, and M. I. Jordan. Learning from dyadic data. In *Advances in Neural Information Processing Systens 11*. MIT Press, 1999.

[8] F.C.N. Pereira, N.Z. Tishby, and L. Lee. Distributional clustering of English words. In *30th Annual Meeting of the Association for Computational Linguistics, Columbus, Ohio*, pages 183–190, 1993.

[9] K. Rose, E. Gurewitz, and G.C. Fox. A deterministic annealing approach to clustering. *Pattern Recognition Letters*, 11(9):589–594, September 1990.

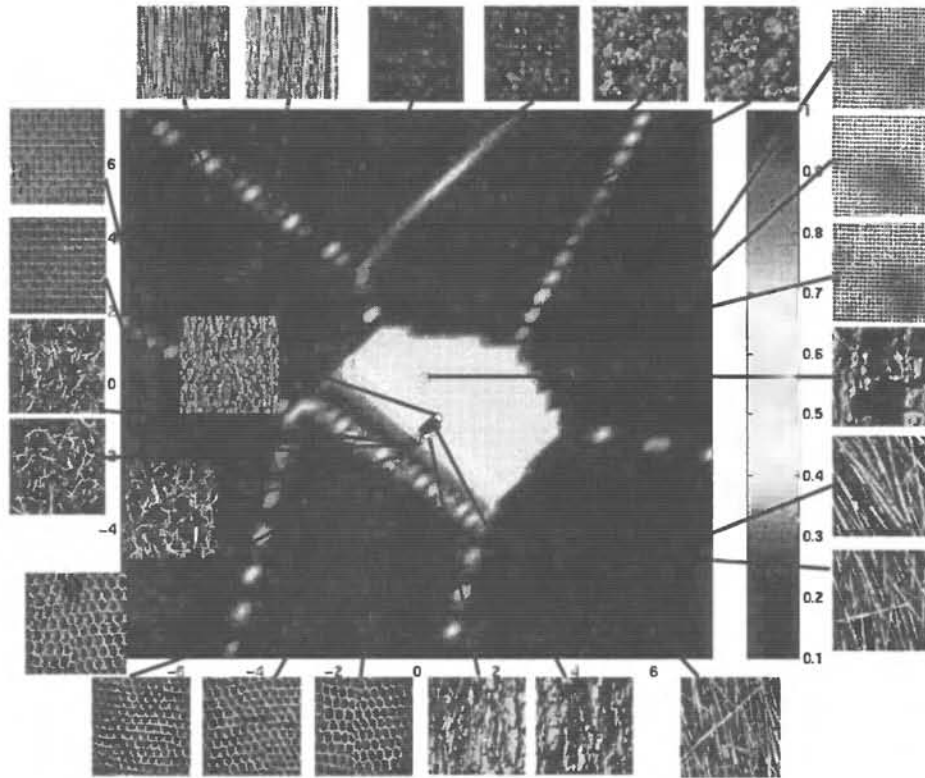

Figure 3: Visualization of a probabilistic grouping structure inferred for a database of 160 Brodatz textures. A mean KL–divergence of 0.031 is obtained.

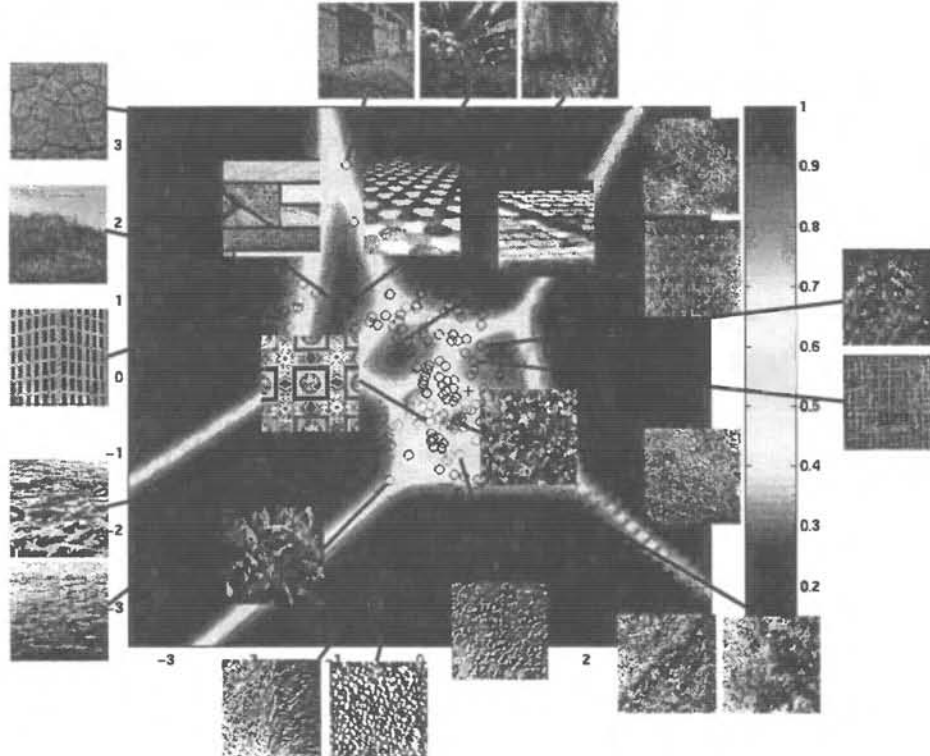

Figure 4: Visualization of a probabilistic grouping structure inferred for 220 images of the VisTex database. A mean KL–divergence of 0.0018 is obtained.